# Neural Network Modeling of Speech and Music Signals

**Axel Röbel**

Technical University Berlin, Einsteinufer 17, Sekr. EN-8, 10587 Berlin, Germany
Tel: +49-30-314 25699, FAX: +49-30-314 21143, email: roebel@kgw.tu-berlin.de

## Abstract

Time series prediction is one of the major applications of neural networks. After a short introduction into the basic theoretical foundations we argue that the iterated prediction of a dynamical system may be interpreted as a model of the system dynamics. By means of RBF neural networks we describe a modeling approach and extend it to be able to model instationary systems. As a practical test for the capabilities of the method we investigate the modeling of musical and speech signals and demonstrate that the model may be used for synthesis of musical and speech signals.

## 1 Introduction

Since the formulation of the reconstruction theorem by Takens [10] it has been clear that a nonlinear predictor of a dynamical system may be directly derived from a systems time series. The method has been investigated extensively and with good success for the prediction of time series of nonlinear systems. Especially the combination of reconstruction techniques and neural networks has shown good results [12].

In our work we extend the ideas of predicting nonlinear systems by the more demanding task of building system models, which are able to resynthesize the systems time series. In the case of chaotic or strange attractors the resynthesis of identical time series is known to be impossible. However, the modeling of the underlying attractor leads to the possibility to resynthesis time series which are consistent with the system dynamics. Moreover, the models may be used for the analysis of the system dynamics, for example the estimation of the Lyapunov exponents [6]. In the following we investigate the modeling of music and speech signals, where the system dynamics are known to be instationary. Therefore, we

develop an extension of the modeling approach, such that we are able to handle instationary systems.

In the following, we first give a short review concerning the state space reconstruction from time series by delay coordinate vectors, a method that has been introduced by Takens [10] and later extended by Sauer et al. [9]. Then we explain the structure of the neural networks we used in the experiments and the enhancements necessary to be able to model instationary dynamics. As an example we apply the neural models to a saxophone tone and a speech signal and demonstrate that the signals may be resynthesized using the neural models. Furthermore, we discuss some of the problems and outline further developments of the application.

## 2  Reconstructing attractors

Assume an $n$-dimensional dynamical system $f(\cdot)$ evolving on an attractor $A$. $A$ has fractal dimension $d$, which often is considerably smaller then $n$. The system state $\vec{z}$ is observed through a sequence of measurements $h(\vec{z})$, resulting in a time series of measurements $y_t = h(\vec{z}(t))$. Under weak assumptions concerning $h(\cdot)$ and $f(\cdot)$ the fractal embedding theorem[9] ensures that, for $D > 2d$, the set of all *delayed coordinate vectors*

$$Y_{D,T} = \{t > t_0 : (y_t, y_{t-T}, \ldots, y_{t-(D-1)T})\}, \tag{1}$$

with an arbitrary delay time $T$, forms an embedding of $A$ in the $D$-dimensional *reconstruction* space. We call the minimal $D$, which yields an embedding of $A$, the *embedding dimension* $D_e$. Because an embedding preserves characteristic features of $A$, especially it is one to one, it may be employed for building a system model. For this purpose the reconstruction of the attractor is used to uniquely identify the systems state thereby establishing the possibility of uniquely predicting the systems evolution. The prediction function may be represented by a hyperplane over the attractor in an $(D + 1)$ dimensional space. By iterating this prediction function we obtain a vector valued system model which, however, is valid only at the respective attractor.

For the reconstruction of instationary systems dynamics we confine ourselves to the case of slowly varying parameters and model the instationary system using a sequence of attractors.

## 3  RBF neural networks

There are different topologies of neural networks that may be employed for time series modeling. In our investigation we used radial basis function networks which have shown considerably better scaling properties, when increasing the number of hidden units, than networks with sigmoid activation function [8]. As proposed by Verleysen et. al [11] we initialize the network using a vector quantization procedure and then apply backpropagation training to finally tune the network parameters. The tuning of the parameters yields an improvement factor of about ten in prediction error compared to the standard RBF network approach [8, 3]. Compared to earlier results [7] the normalization of the hidden layer activations yields a small improvement in the stability of the models.

The resulting network function for $m$-dimensional vector valued output is of the form

$$\vec{N}(\vec{x}) = \sum_j \vec{w}_j \frac{\exp\left(-\left(\frac{\vec{c}_j - \vec{x}}{\sigma_j}\right)^2\right)}{\sum_i exp\left(-\left(\frac{\vec{c}_i - \vec{x}}{\sigma_i}\right)^2\right)} + \vec{b}, \tag{2}$$

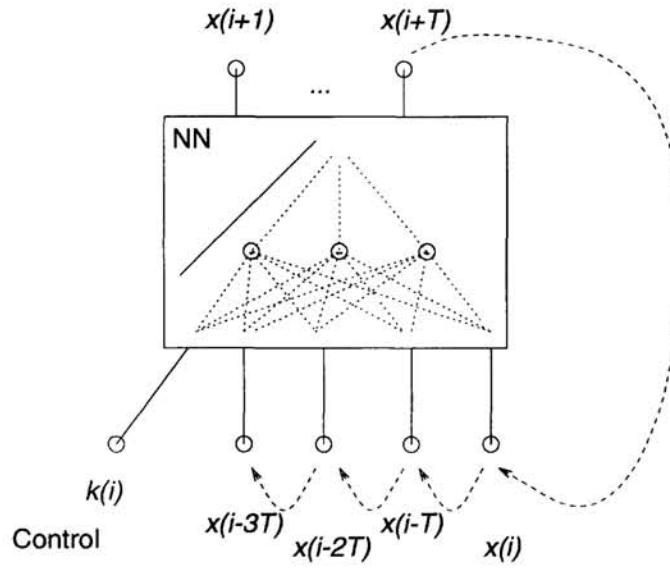

Fig. 1: Input/Output structure of the neural model.

where $\sigma_j$ represents the standard deviation of the Gaussian, the input $\vec{x}$ and the centers $\vec{c}$ are $n$-dimensional vectors and $\vec{b}$ and $\vec{w}_j$ are $m$-dimensional parameters of the network. Networks of the form eq. (2) with a finite number of hidden units are able to approximate arbitrary closely all continuous mappings $R^n \to R^m$ [4]. This universal approximation property is the foundation of using neural networks for time series modeling, where we denote them as *neural models*. In the context of the previous section the neural models are approximating the systems prediction function.

To be able to represent instationary dynamics, we extend the network according to figure 1 to have an additional input, that enables the control of the actual mapping

$$\vec{N}(\vec{x}, k) = \sum_j \vec{w}_j \frac{\exp\left(-\left(\frac{\vec{c}_j - \vec{x}}{\sigma_j}\right)^2 - \left(\frac{t_j - k}{\sigma_{t_j}}\right)^2\right)}{\sum_i \exp\left(-\left(\frac{\vec{c}_i - \vec{x}}{\sigma_i}\right)^2 - \left(\frac{t_i - k}{\sigma_{t_i}}\right)^2\right)} + \vec{b}. \tag{3}$$

This model is close to the Hidden Control Neural Network described in [2]. From the universal approximation properties of the RBF-networks stated above it follows, that eq. (3) with appropriate control sequence $k(i)$ is able to approximate any sequence of functions. In the context of time series prediction the value $i$ represents the actual sample time. The control sequence may be optimized during training, as described in [2], The optimization of $k(i)$ requires prohibitively large computational power if the number of different control values, the domain of $k$ is large. However, as long as the systems instationarity is described by a smooth function of time, we argue that it is possible to select $k(i)$ to be a fixed linear function of $i$. With the preselected $k(i)$ the training of the network adapts the parameters $t_j$ and $\sigma_{tj}$ such that the model evolution closely follows the systems instationarity.

## 4   Neural models

As is shown in figure 1 we use the delayed coordinate vectors and a selected control sequence to train the network to predict the sequence of the following $T$ time samples. The

vector valued prediction avoids the need for a further interpolation of the predicted samples. Otherwise, an interpolation would be necessary to obtain the original sample frequency, but, because the Nyquist frequency is not regarded in choosing $T$, is not straightforward to achieve.

After training we initialize the network input with the first input vector $(\vec{x}_0, k(0))$ of the time series and iterate the network function shifting the network input and using the latest output unit to complete the new input. The control input may be copied from the training phase to resynthesize the training signal or may be varied to emulate another sequence of system dynamics.

The question that has to be posed in this context is concerned with the stability of the model. Due to the prediction error of the model the iteration will soon leave the reconstructed attractor. Because there exists no training data from the neighborhood of the attractor the minimization of the prediction error of the network does not guaranty the stability of the model [5]. Nevertheless, as we will see in the examples, the neural models are stable for at least some parameters $D$ and $T$.

Due to the high density of training data the method for stabilizing dynamical models presented in [5] is difficult to apply in our situation. Another approach to increase the model stability is to lower the gradient of the prediction function for the directions normal to the attractor. This may be obtained by disturbing the network input during training with a small noise level. While conceptually straightforward, we found that this method is only partly successful. While the resulting prediction function is smoother in the neighborhood of the attractor, the prediction error for training with noise is considerably higher as expected from the noise free results, such that the overall effect often is negative. To circumvent the problems of training with noise further investigations will consider a optimization function with regularization that directly penalizes high derivatives of the network with respect to the input units [1]. The stability of the models is a major subject of further research.

## 5   Practical results

We have applied our method to two acoustic time series, a single saxophone tone, consisting of 16000 samples sampled at 32kHz and a speech signal of the word *manna*[1]. The latter time series consists of 23000 samples with a sampling rate of 44.1kHz. Both time series have been normalized to stay within the interval $[-1, 1]$. The estimation of the dimension of the underlying attractors yields a dimension of about 2-3 in both cases.

We chose the control input $k(i)$ to be linear increasing from $-0.8$ to $0.8$. Stable models we found for both time series using $D > 5$. Namely for the parameter $T$ we observed considerable impact on the model quality. While smaller $T$ results in better one step ahead prediction, the iterated model often becomes unstable. This might be explained by the decrease in variation within the prediction hyperplane, that has to be learned. For small $T$ the model tends to become linear and does not capture the nonlinear characteristics of the system. Therefore the iteration of those models failed.

To large values of $T$ results in an insufficient one step ahead prediction error, which pushes the model far away from the attractor also producing unstable behavior.

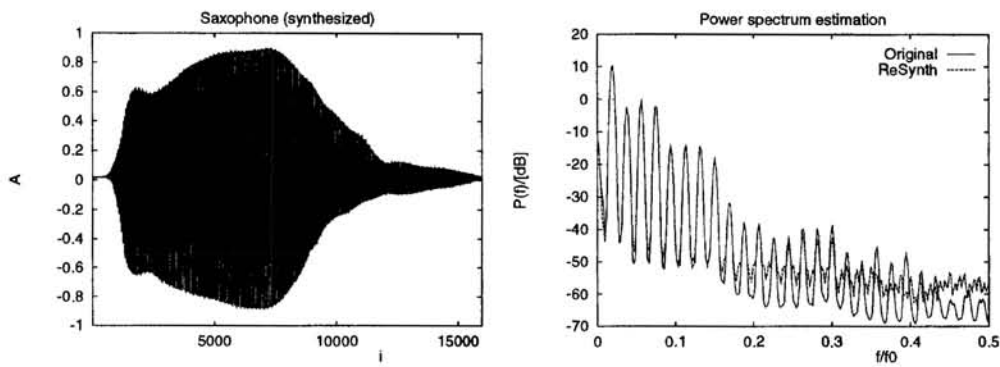

Fig. 2: Synthesized saxophone signal and power spectrum estimation for the original (solid) and synthesized (dashed) signal.

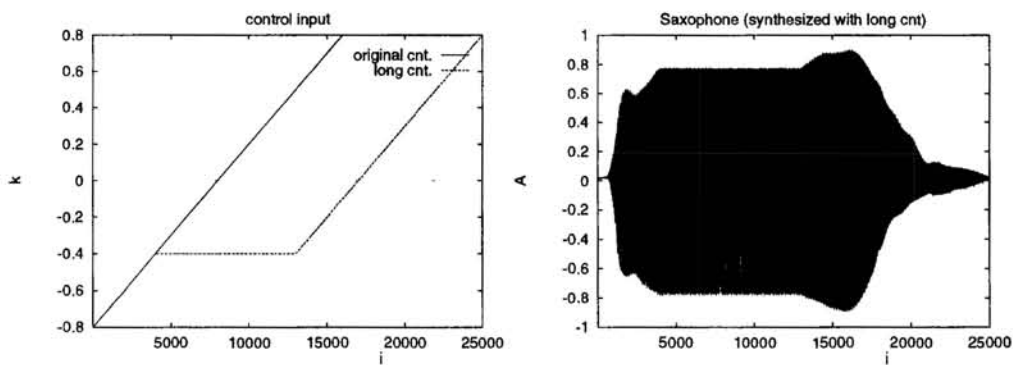

Fig. 3: Varying the synthesized tone by varying the control input sequence.

## 5.1 Modeling a saxophone

In the following we consider the results for the saxophone model. The model we present consists of 10 input units, 200 hidden units and 5 output units and was trained with additional Gaussian noise at the input. The standard deviation of the noise is 0.0005 and the RMS training error obtained is 0.005. The resulting saxophone model is able to resynthesize a signal which is nearly indistinguishable from the original one. The resynthesized time series is shown in figure 2. The time series follows the original one with a small phase shift, which stems from a small difference in the onset of the model. Also in figure 2 the power spectrum of the saxophone signal and the neural model is shown. From the spectrum we see the close resemblance of the sound.

One major demand for the practical application of the proposed musical instrument models is the possibility to control the synthesized sound. At the present state there exists only one control input to the model. Nevertheless, it is interesting to investigate the effect of varying the control input of the model. We tried different control input sequences to synthesize saxophone tones. It turns out that the model remains stable such that we are able to control the envelope of the sound. An example of a tone with increased duration is shown in figure 3. In this example the control input first follows the trained version, then remains constant to produce a longer duration of the tone and then increases to reproduce the decay of the tone from the trained time series.

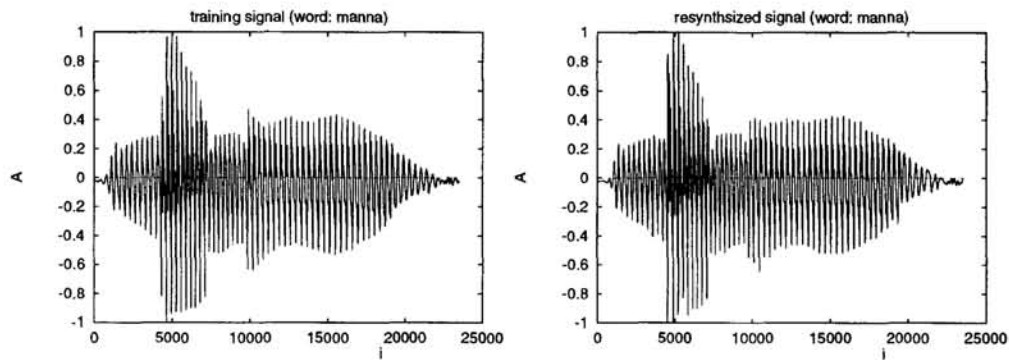

Fig. 4: Original and synthesized signal of the word *manna*.

## 5.2 Modeling a speech signal

For modeling the time series of the spoken word *manna* we used a similar network compared to the saxophone model. Due to the increased instationarity in the signal we needed an increased number of RBF units in the network. The best results up to now has been obtained with a network of 400 hidden units, delay time $T = 8$, output dimension 8 and input dimension 11.

In figure 4 we show the original and the resynthesized signal. The quality of the model is not as high as in the case of the saxophone. Nevertheless, the word is quite understandable. From the figure we see, that the main problems stem from the transitions between consecutive phonemes. These transitions are rather quick in time and, therefore, there exists only a small amount of data describing the dynamics of the transitions. We assume that more training examples of the same word will cure the problem. However, it will probably require a well trained speaker to reproduce the dynamics in speaking the same word twice.

## 6 Further developments

There are two practical applications that directly follow from the presented results. The first one is to synthesize music signals. To consider musicians demands, we need to enhance the control of the synthesized signals. Therefore, in the future we will try to enlarge the models, incorporating different flavors of sound into the same model and adding additional control inputs. Especially we plan to build models for different volume and pitch.
As a second application we will further investigate the possibilities for using the neural models as a speech synthesizer. An interesting topic of further research would be the extension of the model with an intonation control input that incorporates the possibility to synthesize different intonations of the same word from one model.

## 7 Summary

The article describes a methodology to build instationary models from time series of dynamical systems. We give theoretical arguments for the universality of the models and discuss some of the restrictions and actual problems. As practical test for the method we apply the models to the demanding task of the synthesis of musical and speech signals. It is demonstrated that the models are capable to resynthesize the trained signals. At the present

state the envelope and duration of the synthesized signals may be controlled. Intended further developments have been shortly described.

## Footnotes

[1]The name of our parallel computer

## References

[1] C. M. Bishop. Training with noise is equivalent to tikhonov regularization. *Neural Computation*, 7(1):108–116, 1995.

[2] E. Levin. Hidden control neural architecture modelling of nonlinear time varying systems and its applications. *IEEE Transactions on Neural Networks*, 4(2):109–116, 1993.

[3] J. Moody and C. Darken. Fast learning in networks of locally-tuned processing units. *Neural Computation*, 1:281–294, 1989.

[4] J. Park and I. Sandberg. Universal approximation using radial-basis-function networks. *Neural Computation*, 3(2):246–257, 1991.

[5] J. C. Principe and J.-M. Kuo. Dynamic modelling of chaotic time series with neural networks. In G. Tesauro, D. S. Touretzky, and T. Leen, editors, *Neural Information Processing Systems 7 (NIPS 94)*, 1995.

[6] A. Röbel. Neural models for estimating lyapunov exponents and embedding dimension from time series of nonlinear dynamical systems. In *Proceedings of the Intern. Conference on Artificial Neural Networks, ICANN'95, Vol. II*, pages 533–538, Paris, 1995.

[7] A. Röbel. Rbf networks for synthesis of speech and music signals. In *3. Workshop Fuzzy-Neuro-Systeme'95*, Darmstadt, 1995. Deutsche Gesellschaft für Informatik e.V.

[8] A. Röbel. Scaling properties of neural networks for the prediction of time series. In *Proceedings of the 1996 IEEE Workshop on Neural Networks for Signal Processing VI*, 1996.

[9] T. Sauer, J. A. Yorke, and M. Casdagli. Embedology. *Journal of Statistical Physics*, 65(3/4):579–616, 1991.

[10] F. Takens. *Detecting Strange Attractors in Turbulence*, volume 898 of *Lecture Notes in Mathematics (Dynamical Systems and Turbulence, Warwick 1980)*, pages 366–381. D.A. Rand and L.S. Young, Eds. Berlin: Springer, 1981.

[11] M. Verleysen and K. Hlavackova. An optimized RBF network for approximation of functions. In *Proceedings of the European Symposium on Artificial Neural Networks, ESANN'94*, 1994.

[12] A. S. Weigend and N. A. Gershenfeld. *Time Series Prediction: Forecasting the Future and Understanding the Past*. Addison-Wesley Pub. Comp., 1993.
